# Incremental Learning for Visual Tracking

**Jongwoo Lim**[†]   **David Ross**[‡]   **Ruei-Sung Lin**[†]   **Ming-Hsuan Yang**[*]

[†] University of Illinois    [‡] University of Toronto    [*] Honda Research Institute

`jlim1@uiuc.edu  dross@cs.toronto.edu  rlin1@uiuc.edu  myang@honda-ri.com`

## Abstract

Most existing tracking algorithms construct a representation of a target object prior to the tracking task starts, and utilize invariant features to handle appearance variation of the target caused by lighting, pose, and view angle change. In this paper, we present an efficient and effective online algorithm that incrementally learns and adapts a low dimensional eigenspace representation to reflect appearance changes of the target, thereby facilitating the tracking task. Furthermore, our incremental method correctly updates the sample mean and the eigenbasis, whereas existing incremental subspace update methods ignore the fact the sample mean varies over time. The tracking problem is formulated as a state inference problem within a Markov Chain Monte Carlo framework and a particle filter is incorporated for propagating sample distributions over time. Numerous experiments demonstrate the effectiveness of the proposed tracking algorithm in indoor and outdoor environments where the target objects undergo large pose and lighting changes.

## 1   Introduction

The main challenges of visual tracking can be attributed to the difficulty in handling appearance variability of a target object. Intrinsic appearance variabilities include pose variation and shape deformation of a target object, whereas extrinsic illumination change, camera motion, camera viewpoint, and occlusions inevitably cause large appearance variation. Due to the nature of the tracking problem, it is imperative for a tracking algorithm to model such appearance variation.

Here we developed a method that, during visual tracking, constantly and efficiently updates a low dimensional eigenspace representation of the appearance of the target object. The advantages of this adaptive subspace representation are several folds. The eigenspace representation provides a compact notion of the "thing" being tracked rather than treating the target as a set of independent pixels, i.e., "stuff" [1]. The use of an incremental method continually updates the eigenspace to reflect the appearance change caused by intrinsic and extrinsic factors, thereby facilitating the tracking process. To estimate the locations of the target objects in consecutive frames, we used a sampling algorithm with likelihood estimates, which is in direct contrast to other tracking methods that usually solve complex optimization problems using gradient-descent approach.

The proposed method differs from our prior work [14] in several aspects. First, the proposed algorithm does not require any training images of the target object before the tracking task starts. That is, our tracker learns a low dimensional eigenspace representation on-line and incrementally updates it as time progresses (We assume, like most tracking algorithms, that the target region has been initialized in the first frame). Second, we extend our sampling method to incorporate a particle filter so that the sample distributions are propagated over time. Based on the eigenspace model with updates, an effective likelihood estimation function is developed. Third, we extend the R-SVD algorithm [6] so that both the sample mean and eigenbasis are correctly updated as new data arrive. Though there are numerous subspace update algorithms in the literature, only the method by Hall et al. [8] is also able

to update the sample mean. However, their method is based on the addition of a single column (single observation) rather than blocks (a number of observations in our case) and thus is less efficient than ours. While our formulation provides an exact solution, their algorithm gives only approximate updates and thus it may suffer from numerical instability. Finally, the proposed tracker is extended to use a robust error norm for likelihood estimation in the presence of noisy data or partial occlusions, thereby rendering more accurate and robust tracking results.

## 2 Previous Work and Motivation

Black et al. [4] proposed a tracking algorithm using a pre-trained view-based eigenbasis representation and a robust error norm. Instead of relying on the popular brightness constancy working principal, they advocated the use of subspace constancy assumption for visual tracking. Although their algorithm demonstrated excellent empirical results, it requires to build a set of view-based eigenbases before the tracking task starts. Furthermore, their method assumes that certain factors, such as illumination conditions, do not change significantly as the eigenbasis, once constructed, is not updated.

Hager and Belhumeur [7] presented a tracking algorithm to handle the geometry and illumination variations of target objects. Their method extends a gradient-based optical flow algorithm to incorporate research findings in [2] for object tracking under varying illumination conditions. Prior to the tracking task starts, a set of illumination basis needs to be constructed at a fixed pose in order to account for appearance variation of the target due to lighting changes. Consequently, it is not clear whether this method is effective if a target object undergoes changes in illumination with arbitrary pose.

In [9] Isard and Blake developed the Condensation algorithm for contour tracking in which multiple plausible interpretations are propagated over time. Though their probabilistic approach has demonstrated success in tracking contours in clutter, the representation scheme is rather primitive, i.e., curves or splines, and is not updated as the appearance of a target varies due to pose or illumination change.

Mixture models have been used to describe appearance change for motion estimation [3] [10]. In Black et al. [3] four possible causes are identified in a mixture model for estimating appearance change in consecutive frames, and thereby more reliable image motion can be obtained. A more elaborate mixture model with an online EM algorithm was recently proposed by Jepson et al. [10] in which they use three components and wavelet filters to account for appearance changes during tracking. Their method is able to handle variations in pose, illumination and expression. However, their $\mathcal{WSL}$ appearance model treats pixels within the target region independently, and therefore does not have notion of the "thing" being tracked. This may result in modeling background rather than the foreground, and fail to track the target.

In contrast to the eigentracking algorithm [4], our algorithm does not require a training phase but learns the eigenbases on-line during the object tracking process, and constantly updates this representation as the appearance changes due to pose, view angle, and illumination variation. Further, our method uses a particle filter for motion parameter estimation rather than the Gauss-Newton method which often gets stuck in local minima or is distracted by outliers [4]. Our appearance-based model provides a richer description than simple curves or splines as used in [9], and has notion of the "thing" being tracked. In addition, the learned representation can be utilized for other tasks such as object recognition. In this work, an eigenspace representation is learned directly from pixel values within a target object in the image space. Experiments show that good tracking results can be obtained with this representation without resorting to wavelets as used in [10], and better performance can potentially be achieved using wavelet filters. Note also that the view-based eigenspace representation has demonstrated its ability to model appearance of objects at different pose [13], and under different lighting conditions [2].

## 3 Incremental Learning for Tracking

We present the details of the proposed incremental learning algorithm for object tracking in this section.

### 3.1 Incremental Update of Eigenbasis and Mean

The appearance of a target object may change drastically due to intrinsic and extrinsic factors as discussed earlier. Therefore it is important to develop an efficient algorithm to update the eigenspace as the tracking task progresses. Numerous algorithms have been developed to update eigenbasis from a time-varying covariance matrix as more data arrive [6] [8] [11] [5]. However, most methods assume zero mean in updating the eigenbasis except the method by Hall et al. [8] in which they consider the change of the mean when updating eigenbasis as each new datum arrives. Their update algorithm only handles one datum per update and gives approximate results, while our formulation handles multiple data at the same time and renders exact solutions.

We extend the work of the classic R-SVD method [6] in which we update the eigenbasis while taking the shift of the sample mean into account. To the best of our knowledge, this formulation with mean update is new in the literature.

Given a $d \times n$ data matrix $A = \{\mathbf{I}_1, \ldots, \mathbf{I}_n\}$ where each column $\mathbf{I}_i$ is an observation (a $d$-dimensional image vector in this paper), we can compute the singular value decomposition (SVD) of $A$, i.e., $A = U\Sigma V^\top$. When a $d \times m$ matrix $E$ of new observations is available, the R-SVD algorithm efficiently computes the SVD of the matrix $A^{'} = (A|E) = U^{'}\Sigma^{'}V^{'\top}$ based on the SVD of $A$ as follows:

1. Apply QR decomposition to and get orthonormal basis $\tilde{E}$ of $E$, and $U^{'} = (U|\tilde{E})$.

2. Let $V^{'} = \left( \begin{smallmatrix} V & 0 \\ 0 & I_m \end{smallmatrix} \right)$ where $I_m$ is an $m \times m$ identity matrix. It follows then,
$$\Sigma^{'} = U^{'\top}A^{'}V^{'} = \left( \begin{smallmatrix} U^\top \\ \tilde{E}^\top \end{smallmatrix} \right)(A|E)\left( \begin{smallmatrix} V & 0 \\ 0 & I_m \end{smallmatrix} \right) = \left( \begin{smallmatrix} U^\top AV & U^\top E \\ \tilde{E}^\top AV & \tilde{E}^\top E \end{smallmatrix} \right) = \left( \begin{smallmatrix} \Sigma & U^\top E \\ 0 & \tilde{E}^\top E \end{smallmatrix} \right).$$

3. Compute the SVD of $\Sigma^{'} = \tilde{U}\tilde{\Sigma}\tilde{V}^\top$ and the SVD of $A^{'}$ is
$$A^{'} = U^{'}(\tilde{U}\tilde{\Sigma}\tilde{V}^\top)V^{'\top} = (U^{'}\tilde{U})\tilde{\Sigma}(\tilde{V}^\top V^{'\top}).$$

Exploiting the properties of orthonormal bases and block structures, the R-SVD algorithm computes the new eigenbasis efficiently. The computational complexity analysis and more details are described in [6].

One problem with the R-SVD algorithm is that the eigenbasis $U$ is computed from $AA^\top$ with the zero mean assumption. We modify the R-SVD algorithm and compute the eigenbasis with mean update. The following derivation is based on scatter matrix, which is same as covariance matrix except a scalar factor.

**Proposition 1** *Let $\mathcal{I}_p = \{\mathbf{I}_1, \mathbf{I}_2, \ldots, \mathbf{I}_n\}$, $\mathcal{I}_q = \{\mathbf{I}_{n+1}, \mathbf{I}_{n+2}, \ldots, \mathbf{I}_{n+m}\}$, and $\mathcal{I}_r = (\mathcal{I}_p|\mathcal{I}_q)$. Denote the means and scatter matrices of $\mathcal{I}_p$, $\mathcal{I}_q$, $\mathcal{I}_r$ as $\bar{\mathbf{I}}_p$, $\bar{\mathbf{I}}_q$, $\bar{\mathbf{I}}_r$, and $\mathcal{S}_p$, $\mathcal{S}_q$, $\mathcal{S}_r$ respectively, then $\mathcal{S}_r = \mathcal{S}_p + \mathcal{S}_q + \frac{nm}{n+m}(\bar{\mathbf{I}}_q - \bar{\mathbf{I}}_p)(\bar{\mathbf{I}}_q - \bar{\mathbf{I}}_p)^\top$.*

Proof: By definition, $\bar{\mathbf{I}}_r = \frac{n}{n+m}\bar{\mathbf{I}}_p + \frac{m}{n+m}\bar{\mathbf{I}}_q$, $\bar{\mathbf{I}}_p - \bar{\mathbf{I}}_r = \frac{m}{n+m}(\bar{\mathbf{I}}_p - \bar{\mathbf{I}}_q)$; $\bar{\mathbf{I}}_q - \bar{\mathbf{I}}_r = \frac{n}{n+m}(\bar{\mathbf{I}}_q - \bar{\mathbf{I}}_p)$ and,

$$
\begin{aligned}
\mathcal{S}_r &= \sum_{i=1}^{n}(\mathbf{I}_i - \bar{\mathbf{I}}_r)(\mathbf{I}_i - \bar{\mathbf{I}}_r)^\top + \sum_{i=n+1}^{n+m}(\mathbf{I}_i - \bar{\mathbf{I}}_r)(\mathbf{I}_i - \bar{\mathbf{I}}_r)^\top \\
&= \sum_{i=1}^{n}(\mathbf{I}_i - \bar{\mathbf{I}}_p + \bar{\mathbf{I}}_p - \bar{\mathbf{I}}_r)(\mathbf{I}_i - \bar{\mathbf{I}}_p + \bar{\mathbf{I}}_p - \bar{\mathbf{I}}_r)^\top + \\
&\quad \sum_{i=m+1}^{n+m}(\mathbf{I}_i - \bar{\mathbf{I}}_q + \bar{\mathbf{I}}_q - \bar{\mathbf{I}}_r)(\mathbf{I}_i - \bar{\mathbf{I}}_q + \bar{\mathbf{I}}_q - \bar{\mathbf{I}}_r)^\top \\
&= \mathcal{S}_p + n(\bar{\mathbf{I}}_p - \bar{\mathbf{I}}_r)(\bar{\mathbf{I}}_p - \bar{\mathbf{I}}_r)^\top + \mathcal{S}_q + m(\bar{\mathbf{I}}_q - \bar{\mathbf{I}}_r)(\bar{\mathbf{I}}_q - \bar{\mathbf{I}}_r)^\top \\
&= \mathcal{S}_p + \frac{nm^2}{(n+m)^2}(\bar{\mathbf{I}}_p - \bar{\mathbf{I}}_q)(\bar{\mathbf{I}}_p - \bar{\mathbf{I}}_q)^\top + \mathcal{S}_q + \frac{n^2 m}{(n+m)^2}(\bar{\mathbf{I}}_p - \bar{\mathbf{I}}_q)(\bar{\mathbf{I}}_p - \bar{\mathbf{I}}_q)^\top \\
&= \mathcal{S}_p + \mathcal{S}_q + \frac{nm}{n+m}(\bar{\mathbf{I}}_p - \bar{\mathbf{I}}_q)(\bar{\mathbf{I}}_p - \bar{\mathbf{I}}_q)^\top \qquad \qquad \square
\end{aligned}
$$

Let $\hat{\mathcal{I}}_p = \{\mathbf{I}_1 - \bar{\mathbf{I}}_p, \ldots, \mathbf{I}_n - \bar{\mathbf{I}}_p\}$, $\hat{\mathcal{I}}_q = \{\mathbf{I}_{n+1} - \bar{\mathbf{I}}_q, \ldots, \mathbf{I}_{n+m} - \bar{\mathbf{I}}_q\}$, and $\hat{\mathcal{I}}_r = \{\mathbf{I}_1 - \bar{\mathbf{I}}_r, \ldots, \mathbf{I}_{n+m} - \bar{\mathbf{I}}_r\}$, and the SVD of $\hat{\mathcal{I}}_r = U_r \Sigma_r V_r^\top$. Let $\tilde{E} = \left( \hat{\mathcal{I}}_q | \sqrt{\frac{nm}{n+m}} (\bar{\mathbf{I}}_p - \bar{\mathbf{I}}_q) \right)$, and use Proposition 1, $\mathcal{S}_r = (\hat{\mathcal{I}}_p | \tilde{E})(\hat{\mathcal{I}}_p | \tilde{E})^\top$. Therefore, we compute SVD on $(\hat{\mathcal{I}}_p | \tilde{E})$ to get $U_r$. This can be done efficiently by the R-SVD algorithm as described above.

In summary, given the mean $\bar{\mathbf{I}}_p$ and the SVD of existing data $\mathcal{I}_p$, i.e., $U_p \Sigma_p V_p^\top$ and new data $\mathcal{I}_q$, we can compute the the mean $\bar{\mathbf{I}}_r$ and the SVD of $\mathcal{I}_r$, i.e., $U_r \Sigma_r V_r^\top$ easily:

1. Compute $\bar{\mathbf{I}}_r = \frac{n}{n+m}\bar{\mathbf{I}}_p + \frac{m}{n+m}\bar{\mathbf{I}}_q$, and $\tilde{E} = \left( \mathcal{I}_q - \bar{\mathbf{I}}_r \mathbf{1}_{(1 \times m)} | \sqrt{\frac{nm}{n+m}} (\bar{\mathbf{I}}_p - \bar{\mathbf{I}}_q) \right)$.

2. Compute R-SVD with $(U_p \Sigma_p V_p^\top)$ and $\tilde{E}$ to obtain $(U_r \Sigma_r V_r^\top)$.

In numerous vision problems, we can further exploit the low dimensional approximation of image data and put larger weights on the recent observations, or equivalently downweight the contributions of previous observations. For example as the appearance of a target object gradually changes, we may want to put more weights on recent observations in updating the eigenbasis since they are more likely to be similar to the current appearance of the target. The forgetting factor $f$ can be used under this premise as suggested in [11] , i.e., $A' = (fA | E) = (U(f\Sigma)V^\top | E)$ where $A$ and $A'$ are original and weighted data matrices, respectively.

## 3.2 Sequential Inference Model

The visual tracking problem is cast as an inference problem with a Markov model and hidden state variable, where a state variable $\mathbf{X}_t$ describes the affine motion parameters (and thereby the location) of the target at time $t$. Given a set of observed images $\mathcal{I}_t = \{\mathbf{I}_1, \ldots, \mathbf{I}_t\}$. we aim to estimate the value of the hidden state variable $\mathbf{X}_t$. Using Bayes' theorem, we have

$$p(\mathbf{X}_t | \mathcal{I}_t) \propto p(\mathbf{I}_t | \mathbf{X}_t) \int p(\mathbf{X}_t | \mathbf{X}_{t-1}) \, p(\mathbf{X}_{t-1} | \mathcal{I}_{t-1}) \, d\mathbf{X}_{t-1}$$

The tracking process is governed by the observation model $p(\mathbf{I}_t | \mathbf{X}_t)$ where we estimate the likelihood of $\mathbf{X}_t$ observing $\mathbf{I}_t$, and the dynamical model between two states $p(\mathbf{X}_t | \mathbf{X}_{t-1})$. The Condensation algorithm [9], based on factored sampling, approximates an arbitrary distribution of observations with a stochastically generated set of weighted samples. We use a variant of the Condensation algorithm to model the distribution over the object's location, as it evolves over time.

## 3.3 Dynamical and Observation Models

The motion of a target object between two consecutive frames can be approximated by an affine image warping. In this work, we use the six parameters of affine transform to model the state transition from $\mathbf{X}_{t-1}$ to $\mathbf{X}_t$ of a target object being tracked. Let $\mathbf{X}_t = (x_t, y_t, \theta_t, s_t, \alpha_t, \phi_t)$ where $x_t, y_t, \theta_t, s_t, \alpha_t, \phi_t$, denote $x$, $y$ translation, rotation angle, scale, aspect ratio, and skew direction at time $t$. Each parameter in $\mathbf{X}_t$ is modeled independently by a Gaussian distribution around its counterpart in $\mathbf{X}_{t-1}$. That is,

$$p(\mathbf{X}_t | \mathbf{X}_{t-1}) = \mathcal{N}(\mathbf{X}_t; \mathbf{X}_{t-1}, \mathbf{\Psi})$$

where $\mathbf{\Psi}$ is a diagonal covariance matrix whose elements are the corresponding variances of affine parameters, i.e., $\sigma_x^2, \sigma_y^2, \sigma_\theta^2, \sigma_s^2, \sigma_\alpha^2, \sigma_\phi^2$.

Since our goal is to use a representation to model the "thing" that we are tracking, we model the image observations using a probabilistic interpretation of principal component analysis [16]. Given an image patch predicated by $\mathbf{X}_t$, we assume the observed image $\mathbf{I}_t$ was generated from a subspace spanned by $U$ centered at $\boldsymbol{\mu}$. The probability that a sample being generated from the subspace is inversely proportional to the distance $d$ from the sample to the reference point (i.e., center) of the subspace, which can be decomposed into the distance-to-subspace, $d_t$, and the distance-within-subspace from the projected sample

to the subspace center, $d_w$. This distance formulation, based on a orthonormal subspace and its complement space, is similar to [12] in spirit.

The probability of a sample generated from a subspace, $p_{d_t}(\mathbf{I}_t|\mathbf{X}_t)$, is governed by a Gaussian distribution:

$$p_{d_t}(\mathbf{I}_t \,|\, \mathbf{X}_t) \;=\; \mathcal{N}(\mathbf{I}_t \,;\, \boldsymbol{\mu},\, UU^\top + \varepsilon I)$$

where $I$ is an identity matrix, $\boldsymbol{\mu}$ is the mean, and $\varepsilon I$ term corresponds to the additive Gaussian noise in the observation process. It can be shown [15] that the negative exponential distance from $\mathbf{I}_t$ to the subspace spanned by $U$, i.e., $\exp(-||(\mathbf{I}_t - \boldsymbol{\mu}) - UU^\top(\mathbf{I}_t - \boldsymbol{\mu})||^2)$, is proportional to $\mathcal{N}(\mathbf{I}_t; \boldsymbol{\mu}, UU^\top + \varepsilon I)$ as $\varepsilon \to 0$.

Within a subspace, the likelihood of the projected sample can be modeled by the Mahalanobis distance from the mean as follows:

$$p_{d_w}(\mathbf{I}_t \,|\, \mathbf{X}_t) \;=\; \mathcal{N}(\mathbf{I}_t \,;\, \boldsymbol{\mu},\, U\Sigma^{-2}U^\top)$$

where $\boldsymbol{\mu}$ is the center of the subspace and $\Sigma$ is the matrix of singular values corresponding to the columns of $U$. Put together, the likelihood of a sample being generated from the subspace is governed by

$$p(\mathbf{I}_t|\mathbf{X}_t) \;=\; p_{d_t}(\mathbf{I}_t|\mathbf{X}_t)\, p_{d_w}(\mathbf{I}_t|\mathbf{X}_t) \;=\; \mathcal{N}(\mathbf{I}_t; \boldsymbol{\mu}, UU^\top + \varepsilon I)\, \mathcal{N}(\mathbf{I}_t; \boldsymbol{\mu}, U\Sigma^{-2}U^\top) \quad (1)$$

Given a drawn sample $\mathbf{X}_t$ and the corresponding image region $\mathbf{I}_t$, we aim to compute $p(\mathbf{I}_t|\mathbf{X}_t)$ using (1). To minimize the effects of noisy pixels, we utilize a robust error norm [4], $\rho(x, \sigma) = \frac{x^2}{\sigma^2 + x^2}$ instead of the Euclidean norm $d(x) = ||x||^2$, to ignore the "outlier" pixels (i.e., the pixels that are not likely to appear inside the target region given the current eigenspace). We use a method similar to that used in [4] in order to compute $d_t$ and $d_w$. This robust error norm is helpful especially when we use a rectangular region to enclose the target (which inevitably contains some noisy background pixels).

## 4 Experiments

To test the performance of our proposed tracker, we collected a number of videos recorded in indoor and outdoor environments where the targets change pose in different lighting conditions. Each video consists of $320 \times 240$ gray scale images and are recorded at 15 frames per second unless specified otherwise. For the eigenspace representation, each target image region is resized to $32 \times 32$ patch, and the number of eigenvectors used in all experiments is set to 16 though fewer eigenvectors may also work well. Implemented in MATLAB with MEX, our algorithm runs at 4 frames per second on a standard computer with 200 particles. We present some tracking results in this section and more tracking results as well as videos can be found at `http://vision.ucsd.edu/~jwlim/ilt/`.

### 4.1 Experimental Results

Figure 1 shows the tracking results using a challenging sequence recorded with a moving digital camera in which a person moves from a dark room toward a bright area while changing his pose, moving underneath spot lights, changing facial expressions and taking off glasses. All the eigenbases are constructed automatically from scratch and constantly updated to model the appearance of the target object while undergoing appearance changes. Even with the significant camera motion and low frame rate (which makes the motions between frames more significant, or equivalently to tracking fast moving objects), our tracker stays stably on the target throughout the sequence.

The second sequence contains an animal doll moving in different pose, scale, and lighting conditions as shown in Figure 2. Experimental results demonstrate that our tracker is able to follow the target as it undergoes large pose change, cluttered background, and lighting variation. Notice that the non-convex target object is localized with an enclosing rectangular window, and thus it inevitably contains some background pixels in its appearance representation. The robust error norm enables the tracker to ignore background pixels and estimate the target location correctly. The results also show that our algorithm faithfully

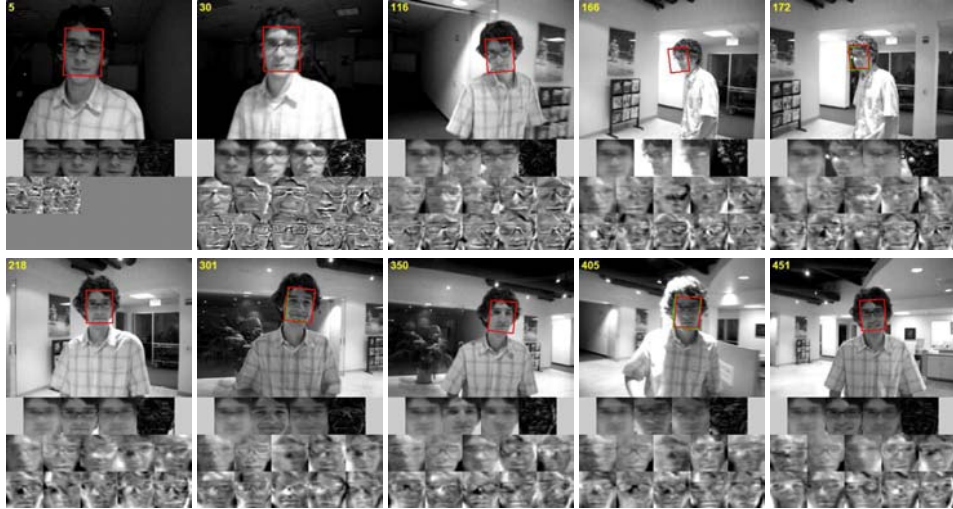

Figure 1: A person moves from dark toward bright area with large lighting and pose changes. The images in the second row shows the current sample mean, tracked region, reconstructed image, and the reconstruction error respectively. The third and forth rows shows 10 largest eigenbases.

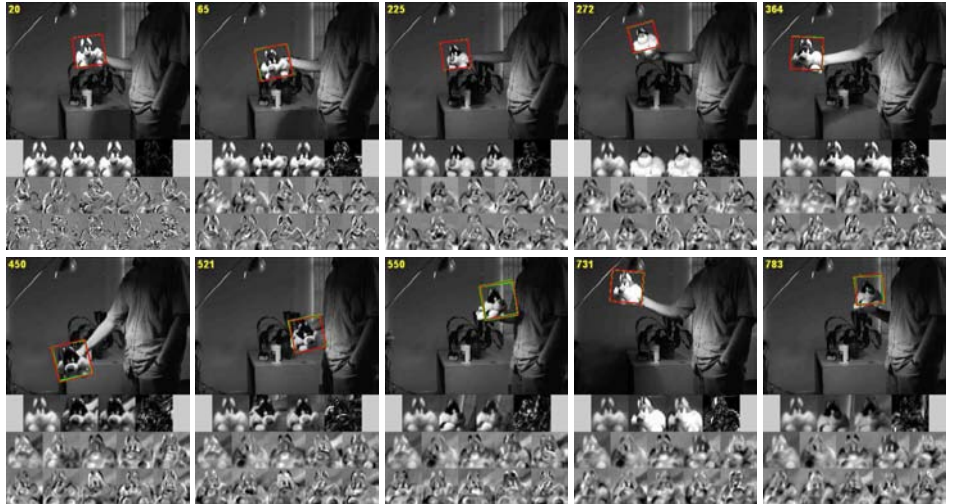

Figure 2: An animal doll moving with large pose, lighting variation in a cluttered background.

models the appearance of the target, as shown in eigenbases and reconstructed images, in the presence of noisy background pixels.

We recorded a sequence to demonstrate that our tracker performs well in outdoor environment where lighting conditions change drastically. The video was acquired when a person walking underneath a trellis covered by vines. As shown in Figure 3, the cast shadow changes the appearance of the target face drastically. Furthermore, the combined pose and lighting variation with low frame rate makes the tracking task extremely difficult. Nevertheless, the results show that our tracker successfully follows the target accurately and robustly. Due to heavy shadows and drastic lighting change, other tracking methods based on gradient, contour, or color information are unlikely to perform well in this case.

## 4.2 Discussion

The success of our tracker can be attributed to several factors. It is well known that the appearance of an object undergoing pose change can be modeled well by view-based

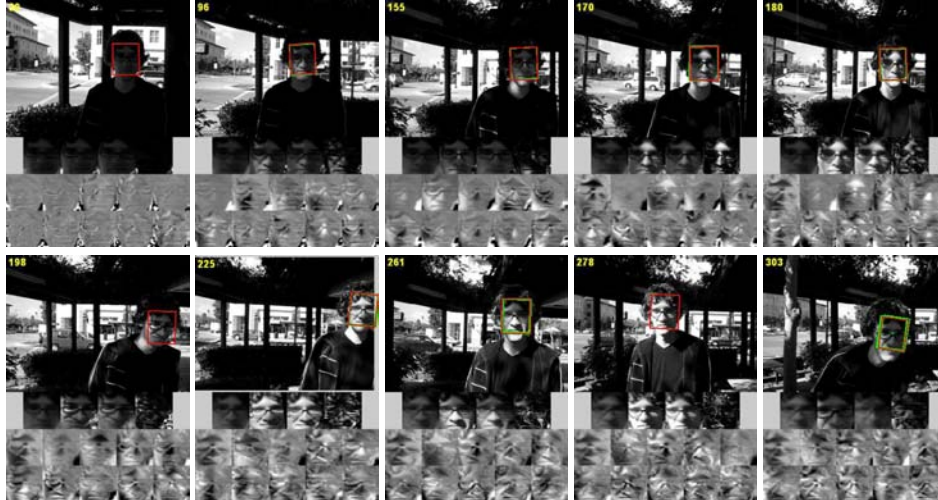

Figure 3: A person moves underneath a trellis with large illumination change and cast shadows while changing his pose. More results can be found in the project web page.

representation [13]. Meanwhile at fixed pose, the appearance of an object in different illumination conditions can be approximated well by a low dimensional subspace [2]. Our empirical results show that these variations can be learned on-line without any prior training phase, and also the changes caused by cast and attached shadows can still be approximated by a linear subspace to some extent. We show a few failure cases at our the web site mentioned earlier. Typically, the failure happens when there is a combination of fast pose change and drastic illumination change.

In this paper, we do not directly address the partial occlusion problems. Empirical results show that temporary and partial occlusions can be handled by our method through the robust error norm and the constant update of the eigenspace. Nevertheless situations arise where we may have prior knowledge of the objects being tracked, and can exploit such information for better occlusion handling.

To demonstrate the potency of our modified R-SVD algorithm in faithfully modeling the object appearance, we compare the reconstructed images using our method and a conventional SVD algorithm. In Figure 4 first row contains a set of images tracked by our tracker, and the second and fourth rows show the reconstructed images using 16 eigenvectors obtained after 121 incremental updates of 605 frame (block size is set to 5), and the top 16 eigenvectors obtained by conventional SVD algorithm using all 605 tracked images. Note that we only maintained 16 eigenvectors during tracking, and discarded the remaining eigenvectors at each update. The residue images are presented in the third and fifth rows, and the average $L2$ reconstruction error per pixel is $5.73 \times 10^{-2}$ and $5.65 \times 10^{-2}$ for our modified R-SVD method and the conventional SVD algorithm respectively. The figure and average reconstruction error shows that our modified R-SVD method is able to effectively model the object appearance without losing detailed information.

## 5   Conclusions and Future Work

We have presented an appearance-based tracker that incrementally learns a low dimensional eigenspace representation for object tracking while the target undergoes pose, illumination and appearance changes. Whereas most tracking algorithms operate on the premise that the object appearance or ambient environment lighting condition does not change as time progresses, our method adapts the model representation to reflect appearance variation of the target, thereby facilitating the tracking task. In contrast to the existing incremental subspace methods, our R-SVD method updates the mean and eigenbasis accurately and

efficiently, and thereby learns a good eigenspace representation to faithfully model the appearance of the target being tracked. Our experiments demonstrate the effectiveness of the proposed tracker in indoor and outdoor environments where the target objects undergo large pose and lighting changes.

The current dynamical model in our sampling method is based on a Gaussian distribution, but the dynamics could be learned from exemplars for more efficient parameter estimation. Our algorithm can be extended to construct a set of eigenbases for modeling nonlinear aspects of appearance variation more precisely and automatically. We aim to address these issues in our future work.

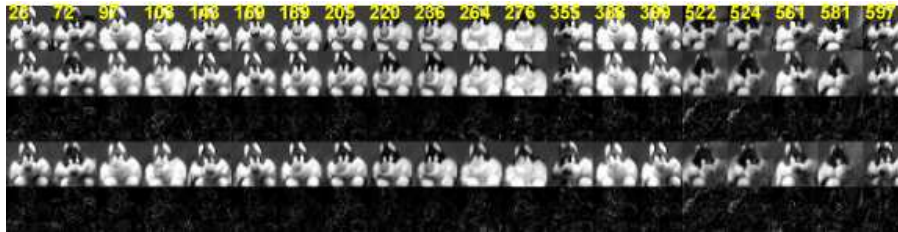

Figure 4: Reconstructed images and errors using our and the conventional SVD algorithms.

# References

[1] E. H. Adelson and J. R. Bergen. The plenoptic function and the elements of early vision. In M. Landy and J. A. Movshon, editors, *Computational Models of Visual Processing*, pp. 1–20. MIT Press, 1991.

[2] P. Belhumeur and D. Kreigman. What is the set of images of an object under all possible lighting conditions. In *Proceedings of IEEE Conference on Computer Vision and Pattern Recognition*, pp. 270–277, 1997.

[3] M. J. Black, D. J. Fleet, and Y. Yacoob. A framework for modeling appearance change in image sequence. In *Proceedings of the Sixth IEEE International Conference on Computer Vision*, pp. 660–667, 1998.

[4] M. J. Black and A. D. Jepson. Eigentracking: Robust matching and tracking of articulated objects using view-based representation. In *Proceedings of European Conference on Computer Vision*, pp. 329–342, 1996.

[5] M. Brand. Incremental singular value decomposition of uncertain data with missing values. In *Proceedings of the Seventh European Conference on Computer Vision*, volume 4, pp. 707–720, 2002.

[6] G. H. Golub and C. F. Van Loan. *Matrix Computations*. The Johns Hopkins University Press, 1996.

[7] G. Hager and P. Belhumeur. Real-time tracking of image regions with changes in geometry and illumination. In *Proceedings of IEEE Conference on Computer Vision and Pattern Recognition*, pp. 403–410, 1996.

[8] P. Hall, D. Marshall, and R. Martin. Incremental eigenanalysis for classification. In *Proceedings of British Machine Vision Conference*, pp. 286–295, 1998.

[9] M. Isard and A. Blake. Contour tracking by stochastic propagation of conditional density. In *Proceedings of the Fourth European Conference on Computer Vision*, volume 2, pp. 343–356, 1996.

[10] A. D. Jepson, D. J. Fleet, and T. F. El-Maraghi. Robust online appearance models for visual tracking. In *Proceedings of IEEE Conference on Computer Vision and Pattern Recognition*, volume 1, pp. 415–422, 2001.

[11] A. Levy and M. Lindenbaum. Sequential Karhunen-Loeve basis extraction and its application to images. *IEEE Transactions on Image Processing*, 9(8):1371–1374, 2000.

[12] B. Moghaddam and A. Pentland. Probabilistic visual learning for object recognition. *IEEE Transactions on Pattern Analysis and Machine Intelligence*, 19(7):696–710, 1997.

[13] H. Murase and S. Nayar. Visual learning and recognition of 3d objects from appearance. *International Journal of Computer Vision*, 14(1):5–24, 1995.

[14] D. Ross, J. Lim, and M.-H. Yang. Adaptive probabilistic visual tracking with incremental subspace update. In *Proceedings of the Eighth European Conference on Computer Vision*, volume 2, pp. 470–482, 2004.

[15] S. Roweis. EM algorithms for PCA and SPCA. In *Advances in Neural Information Processing Systems 10*, pp. 626–632, 1997.

[16] M. E. Tipping and C. M. Bishop. Probabilistic principal component analysis. *Journal of the Royal Statistical Society, Series B*, 61(3):611–622, 1999.
